# Modeling Memory Transfer and Savings in Cerebellar Motor Learning

**Naoki Masuda**
RIKEN Brain Science Institute
Wako, Saitama 351-0198, Japan
masuda@brain.riken.jp

**Shun-ichi Amari**
RIKEN Brain Science Institute
Wako, Saitama 351-0198, Japan
amari@brain.riken.jp

## Abstract

There is a long-standing controversy on the site of the cerebellar motor learning. Different theories and experimental results suggest that either the cerebellar flocculus or the brainstem learns the task and stores the memory. With a dynamical system approach, we clarify the mechanism of transferring the memory generated in the flocculus to the brainstem and that of so-called savings phenomena. The brainstem learning must comply with a sort of Hebbian rule depending on Purkinje-cell activities. In contrast to earlier numerical models, our model is simple but it accommodates explanations and predictions of experimental situations as qualitative features of trajectories in the phase space of synaptic weights, without fine parameter tuning.

## 1 Introduction

The cerebellum is involved in various types of motor learning. As schematically shown in Fig. 1, the cerebellum is composed of the cerebellar cortex and the cerebellar nuclei (we depict the vestibular nucleus $VN$ in Fig. 1). There are two main pathways linking external input from the mossy fibers ($mf$) to motor outputs, which originate from the cerebellar nuclei. The pathway that relays the mossy fibers directly to the cerebellar nuclei is called the direct pathway. Each nucleus cell receives about $10^4$ mossy fiber synapses.

The pathway involving the mossy fibers, the granule cells ($gr$), the parallel fibers ($pl$), and the Purkinje cells ($Pr$) in the flocculo-nodular lobes of the cerebellar cortex, is called the indirect pathway. Because the Purkinje cells, which are the sole source of output from the cerebellar cortex, are GABAergic, firing rates of the nuclei are suppressed when this pathway is active. The indirect pathway also includes recurrent collaterals terminating on various types of inhibitory cells. Another anatomical feature of the indirect pathway is that climbing fibers ($Cm$ in Fig. 1) from the inferior olive ($IO$) innervate on Purkinje cells. Taking into account the huge mass of intermediate computational units in the indirect pathway, or the granule cells, Marr conjectured that the cerebellum operates as a perceptron with high computational power [8]. The climbing fibers were thought to induce long-term potentiation (LTP) of $pl$-$Pr$ synapses to reinforce the signal transduction. Albus claimed that long-term depression (LTD) rather than LTP should occur so that the Purkinje cells inhibit the nuclei [2]. The climbing fibers were thought to serve as teaching lines that convey error-correcting signals.

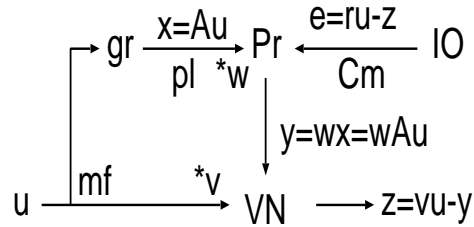

Figure 1: Architecture of the VOR model.

The vestibulo-ocular reflex (VOR) is a standard benchmark for exploring synaptic substrates of cerebellar motor learning. The VOR is a short-latency reflex eye movement that stabilizes images on the retina during head movement. Motion of the head drives eye movements in the opposite direction. When a subject wears a prism, adaptation of the VOR gain occurs for image stabilization. In this context, *in vivo* experiments confirmed that the LTD hypothesis is correct (reviewed in [6]). However, the cerebellum is not the only site of convergence of visual and vestibular signals. The learning scheme depending only on the indirect pathway is called the flocculus hypothesis. An alternative is the brainstem hypothesis in which synaptic plasticity is assumed to occur in the direct pathway ($mf \rightarrow VN$) [12]. This idea is supported by experimental evidence that flocculus shutdown after 3 days of VOR adaptation does not impair the motor memory [7]. Moreover, in other experiments, plasticity of the Purkinje cells in response to vestibular inputs, as required in the flocculus hypothesis, really occurs but in the direction opposite to that predicted by the flocculus hypothesis [5, 12]. Also, LTP of the $mf$-$VN$ synapses, which is necessary to implement the brainstem hypothesis [3], has been suggested in experiments [14].

Relative contributions of the flocculus mechanism and the brainstem mechanism to motor learning remain illusive [3, 5, 9]. The same controversy exists regarding the mechanism of associative eyelid conditioning [9, 10, 11]. Related is the distinction between short-term and long-term plasticities. Many of the experiments in favor of the flocculus hypothesis are concerned with short-term learning, whereas plasticity involving the vestibular nuclei is suggested to be functional in the long term. Short-term motor memory in the flocculus may eventually be transferred to the brainstem. This is termed the memory transfer hypothesis [9]. Medina and Mauk proposed a numerical model and examined what types of brainstem learning rules are compatible with memory transfer [10]. They concluded that the brainstem plasticity should be driven by coincident activities of the Purkinje cells and the mossy fibers. The necessity of Hebbian type of learning in the direct pathway is also supported by another numerical model [13]. We propose a much simpler model to understand the essential mechanism of memory transfer without fine parameter manipulations.

Another goal of this work is to explain savings of learning. Savings are observed in natural learning tasks. Because animals can be trained just for a limited amount of time per day, the task period and the rest period, of e.g. 1 day, alternate. Performance is improved during the task period, and it degrades during the rest period (in the dark). However, when the alternation is repeated, the performance is enhanced more rapidly and progressively in later sessions [7] (also, S. Nagao, private communication). The flocculus may be responsible for daily rapid learning and forgetting, and the brainstem may underlie gradual memory consolidation [11]. While our target phenomenon of interest is the VOR, the proposed model is fairly general.

## 2 Model

Looking at Fig. 1, let us denote by $\mathbf{u} \in \mathbf{R}^m$ the external input to the mossy fibers. It is propagated to the granule cells via synaptic connectivity represented by an $n$ by $m$ matrix $A$, where presumably $n \gg m$. The output of the granule cells, or $\mathbf{x} \equiv A\mathbf{u} \in \mathbf{R}^n$, is received by the Purkinje-cell layer. For simplicity, we assume just one Purkinje cell whose output is written as $y \equiv \mathbf{w}x$, where $\mathbf{w} \in \mathbf{R} \times \mathbf{R}^m$. Since $pl\text{-}Pr$ synapses are excitatory, the elements of $\mathbf{w}$ are positive. The direct pathway ($mf \to VN$) is defined by a plastic connection matrix $\mathbf{v} \in \mathbf{R} \times \mathbf{R}^m$. The output to the VOR actuator is given by $z = \mathbf{v}\mathbf{u} - y = \mathbf{v}\mathbf{u} - \mathbf{w}A\mathbf{u}$, which is the output of the sole neuron of the cerebellar nuclei. This form of $z$ takes into account that the contribution of the indirect pathway is inhibitory and that of the direct pathway is excitatory.

The animal learns to adapt $z$ as close as possible to the desirable motor output $\mathbf{r}\mathbf{u}$. For a large (resp. small) desirable gain $\mathbf{r}$, the correct direction of synaptic changes is the decrease (resp. increase) in $\mathbf{w}$ and the increase (resp. decrease) in $\mathbf{v}$ [5]. The learning error $e \equiv \mathbf{r}\mathbf{u} - z$ is carried by the climbing fibers and projects onto the Purkinje cell, which enables supervised learning [6]. The LTD of $\mathbf{w}$ occurs when the parallel-fiber input and the climbing-fiber input are simultaneously large [6, 9]. Since we can write

$$\dot{\mathbf{w}} = -\eta_1 e\mathbf{x} = -\frac{1}{2}\eta_1 \frac{\partial e^2}{\partial \mathbf{w}}, \tag{1}$$

where $\eta_1$ is the learning rate, $\mathbf{w}$ evolves to minimize $e^2$. Equation (1) is a type of Widrow-Hoff rule [4, p. 320]. With spontaneous inputs only, or in the presence of $\mathbf{x}$ and the absence of $e$, $\mathbf{w}$ experiences LTP [6, 9]. We model this effect by adding $\eta_2\mathbf{x}$ to Eq. (1). This term provides subtractive normalization that counteracts the use-dependent LTD [4, p. 290]. However, subtractive normalization cannot prohibit $\mathbf{w}$ from running away when the error signal is turned off. Therefore, we additionally assume multiplicative normalization term $\eta_3\mathbf{w}$ to limit the magnitude of $\mathbf{w}$ [4, p. 290, 314]. In the end, Eq. (1) is modified to

$$\dot{\mathbf{w}} = -\eta_1(\mathbf{r}\mathbf{u} - \mathbf{v}\mathbf{u} + \mathbf{w}A\mathbf{u})A\mathbf{u} + \eta_2 A\mathbf{u} - \eta_3\mathbf{w}, \tag{2}$$

where $\eta_2$ and $\eta_3$ are rates of memory decay satisfying $\eta_2, \eta_3 \ll \eta_1$.

In the dark, the VOR gain, which might have changed via adaptation, tends back to a value close to unity [5]. Let us represent this reference gain by $\mathbf{r} = \mathbf{r}_0$. With the synaptic strengths in this null condition denoted by $(\mathbf{w}, \mathbf{v}) = (\mathbf{w}_0, \mathbf{v}_0)$, we obtain $\mathbf{r}_0\mathbf{u} = \mathbf{v}_0\mathbf{u} - \mathbf{w}_0 A\mathbf{u}$. By setting $\dot{\mathbf{w}} = 0$ in Eq. (2), we derive

$$\eta_2 A\mathbf{u} = \eta_1(\mathbf{r}_0\mathbf{u} - \mathbf{v}_0\mathbf{u} + \mathbf{w}_0 A\mathbf{u})A\mathbf{u} + \eta_3\mathbf{w}_0 = \eta_3\mathbf{w}_0. \tag{3}$$

Substituting Eq. (3) into Eq. (2) results in

$$\dot{\mathbf{w}} = -\eta_1 \left(\mathbf{r}\mathbf{u} - \mathbf{v}\mathbf{u} + \mathbf{w}A\mathbf{u}\right)A\mathbf{u} - \eta_3 \left(\mathbf{w} - \mathbf{w}_0\right). \tag{4}$$

Experiments show that $\mathbf{v}$ can be potentiated [14]. Enhancement of the excitability of the nucleus output ($z$) in response to tetanic stimulation, or sustained $\mathbf{u}$, is also in line with the LTP of $v$ [1]. In contrast, LTD of $\mathbf{v}$ is biologically unknown. Numerical models suggest that LTP in the nuclei should be driven by $y$ [10, 11]. However, the mechanism and the specificity underlying plasticity of $\mathbf{v}$ are not well understood [9]. Therefore, we assume that both LTP and LTD of $\mathbf{v}$ occur in an associative manner, and we represent the LTP effect by a general function $F$. In parallel to the learning rule of $\mathbf{w}$, we assume a subtractive normalization term $-\eta_5\mathbf{u}$ [10]. We also add a multiplicative normalization term $\eta_6\mathbf{v}$ to constrain $\mathbf{v}$. Finally, we obtain

$$\dot{\mathbf{v}} = \eta_4\mathbf{F}(\mathbf{u}, y, z, e) - \eta_5\mathbf{u} - \eta_6\mathbf{v}. \tag{5}$$

Presumably, $\mathbf{v}$ changes much more slowly (on a time scale of 8–12 hr) than $\mathbf{w}$ changes (0.5 hr) [10, 13]. Therefore, we assume $\eta_1 \gg \eta_4 \gg \eta_5, \eta_6$.

## 3 Analysis of Memory Transfer

Let us examine a couple of learning rules in the direct pathway to identify robust learning mechanisms.

### 3.1 Supervised learning

Although the climbing fibers carrying $e$ send excitatory collaterals to the cerebellar nuclei, supervised learning there has very little experimental support [5]. Here we show that supervised learning in the direct pathway is theoretically unlikely. Let us assume that modification of $\mathbf{v}$ decreases $|e|$. Accordingly, we set $\mathbf{F} = -\partial e^2/\partial \mathbf{v} = e\mathbf{u}$. Then, Eq. (5) becomes

$$\dot{\mathbf{v}} = \eta_4(r\mathbf{u} - \mathbf{v}\mathbf{u} + \mathbf{w}A\mathbf{u})\mathbf{u} - \eta_5\mathbf{u} - \eta_6\mathbf{v}. \tag{6}$$

In the natural situation, $\mathbf{r} = \mathbf{r}_0$. Hence,

$$\eta_5\mathbf{u} = \eta_4(r_0\mathbf{u} - \mathbf{v}_0\mathbf{u} + \mathbf{w}_0 A\mathbf{u})\mathbf{u} - \eta_6\mathbf{v}_0 = -\eta_6\mathbf{v}_0. \tag{7}$$

Inserting Eq. (7) into Eq. (6) yields

$$\dot{\mathbf{v}} = \eta_4\left(r\mathbf{u} - \mathbf{v}\mathbf{u} + \mathbf{w}A\mathbf{u}\right)\mathbf{u} - \eta_6(\mathbf{v} - \mathbf{v}_0). \tag{8}$$

For further analysis, let us assume $m = n = 1$ (for which we quit bold notations) and perform the slow-fast analysis based on $\eta_1 \gg \eta_3, \eta_4 \gg \eta_6$. Equations (4) and (8) define the nullclines $\dot{w} = 0$ and $\dot{v} = 0$, which are represented respectively by

$$v = v_0 + r - r_0 + \frac{\eta_1 A^2 u^2 + \eta_3}{\eta_1 A u^2}(w - w_0), \quad \text{and} \tag{9}$$

$$v = v_0 + \frac{\eta_4 u^2}{\eta_4 u^2 + \eta_6}(r - r_0) + \frac{\eta_4 A u^2}{\eta_4 u^2 + \eta_6}(w - w_0). \tag{10}$$

Since $\dot{w} = O(\eta_1) \gg O(\eta_4) = \dot{v}$ in an early stage, a trajectory in the $w$-$v$ plane initially approaches the fast manifold (Eq. (9)) and moves along it toward the equilibrium given by

$$w^* = w_0 - \frac{\eta_1\eta_6 A u^2(r - r_0)}{\eta_1\eta_6 A^2 u^2 + \eta_3\eta_4 u^2 + \eta_3\eta_6}, \quad v^* = v_0 + \frac{\eta_3\eta_4 u^2(r - r_0)}{\eta_1\eta_6 A^2 u^2 + \eta_3\eta_4 u^2 + \eta_3\eta_6}. \tag{11}$$

LTD of $w$ and LTP of $v$ are expected for adaptation to a larger gain ($r > r_0$), and LTP of $w$ and LTD of $v$ are expected for $r < r_0$. The results are consistent with both the flocculus hypothesis and the brainstem hypothesis as far as the direction of learning is concerned [5]. When $r > r_0$ (resp. $r < r_0$), LTD (resp. LTP) of $w$ first occurs to decrease the learning error. Then, the motor memory stored in $w$ is gradually transferred by LTP (resp. LTD) of $v$ replacing LTD (resp. LTP) of $w$. In the long run, the memory is stored mainly in $v$, not in $w$.

However, the memory transfer based on supervised learning has fundamental deficiencies. First, since $\eta_1 \gg \eta_3$ and $\eta_4 \gg \eta_6$, both nullclines Eqs. (9) and (10) have a slope close to $A$ in the $w$-$v$ plane. This means that the relative position of the equilibrium depends heavily on the parameter values, especially on the learning rates, the choice of which is rather arbitrary. Then, $(w^*, v^*)$ may be located so that, for example, the LTP of $w$ or LTD of $v$ results from $r > r_0$. Also, the degree of transfer, or $|w^* - w_0| / |v^* - v_0|$, is not robust against parameter changes. This may underlie the fact that LTD of $w$ was not followed by partial LTP in the numerical simulations in [10]. Even if the position of $(w^*, v^*)$ happens to support LTD of $w$ and LTP of $v$, memory transfer takes a long time. This is because Eqs. (9) and (10) are fairly close, which means that $\dot{v}$ is small on the fast manifold ($\dot{w} = 0$).

We can also imagine a type of Hebbian rule with $F = \partial z^2/\partial \mathbf{v} = z\mathbf{u}$. Similar calculations show that this rule also realizes memory transfer only in an unreliable manner.

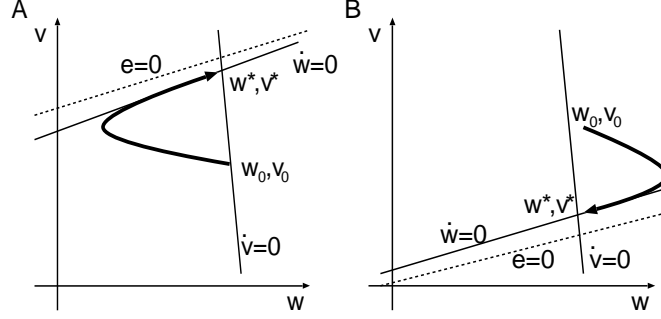

Figure 2: Dynamics of the synaptic weights in the Purkinje cell-dependent learning. (A) $r > r_0$ and (B) $r < r_0$.

## 3.2 Purkinje cell-dependent learning

Results of numerical studies support that $\mathbf{v}$ should be subject to a type of Hebbian learning depending on two afferents to the vestibular nuclei, namely, $\mathbf{u}$ and $y$ [10, 11, 13]. Changes in the VOR gain are signaled by $y$. Since LTP should logically occur when $y$ is small and $\mathbf{u}$ is large, we set $\mathbf{F} = (y_{max} - y)\mathbf{u}$, where $y_{max}$ is the maximum firing rate of the Purkinje cell. Then, we obtain

$$\dot{\mathbf{v}} = \eta_4(y_{max} - \mathbf{w}A\mathbf{u})\mathbf{u} - \eta_5\mathbf{u} - \eta_6\mathbf{v}. \tag{12}$$

The subtraction normalization is determined from the equilibrum condition:

$$\eta_5\mathbf{u} = \eta_4(y_{max} - \mathbf{w}_0A\mathbf{u}) - \eta_6\mathbf{v}_0. \tag{13}$$

Substituting Eq. (13) into Eq. (12) yields

$$\dot{\mathbf{v}} = \eta_4(\mathbf{w}_0 - \mathbf{w})A\mathbf{u}^2 + \eta_6(\mathbf{v}_0 - \mathbf{v}). \tag{14}$$

When $m = n = 1$, the nullclines are given by Eq. (9) and

$$v = v_0 - \frac{\eta_4 Au^2}{\eta_6}(w - w_0), \tag{15}$$

which are depicted in Fig. 2(A) and (B) for $r > r_0$ and $r < r_0$, respectively. As shown by arrows in Fig. 2, trajectories in the $w$-$v$ space first approach the fast manifold Eq. (9) and then move along it toward the equilibrium given by

$$w^* = w_0 - \frac{\eta_1\eta_6 Au^2(r - r_0)}{\eta_1\eta_4 A^2u^4 + \eta_1\eta_6 A^2u^2 + \eta_3\eta_6}, \quad v^* = v_0 + \frac{\eta_1\eta_4 A^2u^4(r - r_0)}{\eta_1\eta_4 A^2u^4 + \eta_1\eta_6 A^2u^2 + \eta_3\eta_6}. \tag{16}$$

Equation (15) has a large negative slope because $\eta_4 \gg \eta_6$. Consequently, setting $r > r_0$ (resp. $r < r_0$) duly results in LTD (resp. LTP) of $w$ and LTP (resp. LTD) of $v$. At the same time, LTD (resp. LTP) of $w$ in an early stage of learning is partially compensated by subsequent LTP (resp. LTD) of $w$, which agrees with previously reported numerical results [10]. In contrast to the supervised and Hebbian learning rules, this learning is robust against parameter changes since the positions and the slopes of the two nullclines are apart from each other. Owing to this property, in the long term, the memory is transferred more rapidly along the $w$-nullcline than for the other two learning rules. Another benefit of the large negative slope of Eq. (15) is that $|v^* - v_0| \gg |w^* - w_0|$ holds, which means efficient memory transfer from $w$ to $v$.

The error at the equilibrum state is

$$e^* = \frac{\eta_3\eta_6(r - r_0)u}{\eta_1\eta_4 A^2 u^4 + \eta_1\eta_6 A^2 u^2 + \eta_3\eta_6}.$$

(17)

Equation (17) guarantees that the $e = 0$ line is located as shown in Fig. 2, and the learning proceeds so as to decrease $|e|$. The performance overshoot, which is unrealistic, does not occur.

## 4  Numerical Simulations of Savings

The learning rule proposed in Sec. 3.2 explains savings as well. To show this, we mimic a situation of savings by periodically alternating the task period and the rest period. Specifically, we start with $r = r_0 = 1$, $w = w_0$, $v = v_0$, and the learning condition ($r = 2$ or $r = 0.5$) is applied for 4 hours a day. During the rest of the day (20 hours), the dark condition is simulated by giving no teaching signal to the model. Changes in the VOR gains for 8 consecutive days are shown in Fig. 3(A) and (C) for $r = 2$ and $r = 0.5$, respectively. The numerical results are consistent with the savings found in other reported experiments [7] and models [11]; the animal forgets much of the acquired gain in the dark, while a small fraction is transferred each day to the cerebellar nuclei. The time-dependent synaptic weights are shown in Fig. 3(B) ($r = 2$) and (D) ($r = 0.5$) and suggest that $v$ is really responsible for savings and that its plasticity needs guidance under the short-term learning of $w$. The memory transfer occurs even in the dark condition, as indicated by the increase (resp. decrease) of $v$ in the dark shown in Fig. 3(B) (resp. (D)). This happens because ruin of the short-term memory of $w$ drives the learning of $v$ for some time even after the daily training has finished. For the indirect pathway, a dark condition defines an off-task period during which $w$ gradually loses its associations.

For comparison, let us deal with the case in which $v$ is fixed. Then, the learning rule Eq. (4) is reduced to

$$\dot{w} = -\eta_1 \left[ (r - r_0)\, u + (w - w_0)\, Au \right] Au - \eta_3 \left( w - w_0 \right).$$

(18)

The VOR adaptation with this rule is shown in Fig. 4(A) ($r = 2$) and (B) ($r = 0.5$). Long-term retention of the acquired gain is now impossible, whereas the short-term learning, or the adaptation within a day, deteriorates little. Since savings do not occur, the ultimate learning error is larger than when $v$ is plastic.

However, if $w$ is fixed and $v$ is plastic, the VOR gain is not adaptive, since $y$ does not carry teaching signals any longer. In this case, we must implement supervised learning of $v$ for learning to occur. Then, $r$ adapts only gradually on the slow time scale of $\eta_4$, and the short-term learning is lost.

## 5  Discussion

Our model explains how the flocculus and the brainstem cooperate in motor learning. Presumably, the indirect pathway involving the flocculus is computationally powerful because of a huge number of intermediate granule cells, but its memory is of short-term nature. The direct pathway bypassing the mossy fibers to the cerebellar nuclei is likely to have less computational power but stores motor memory for a long period. A part of the motor memory is expected to be passed from the flocculus to the nuclei. This happens in a robust manner if the direct pathway is equipped with the learning rule dependent on correlation between the Purkinje-cell firing and the mossy-fiber firing. To explore whether associative LTP/LTD in the cerebellar nuclei really exists will be a subject of future experimental work. Our model is also applicable to savings.

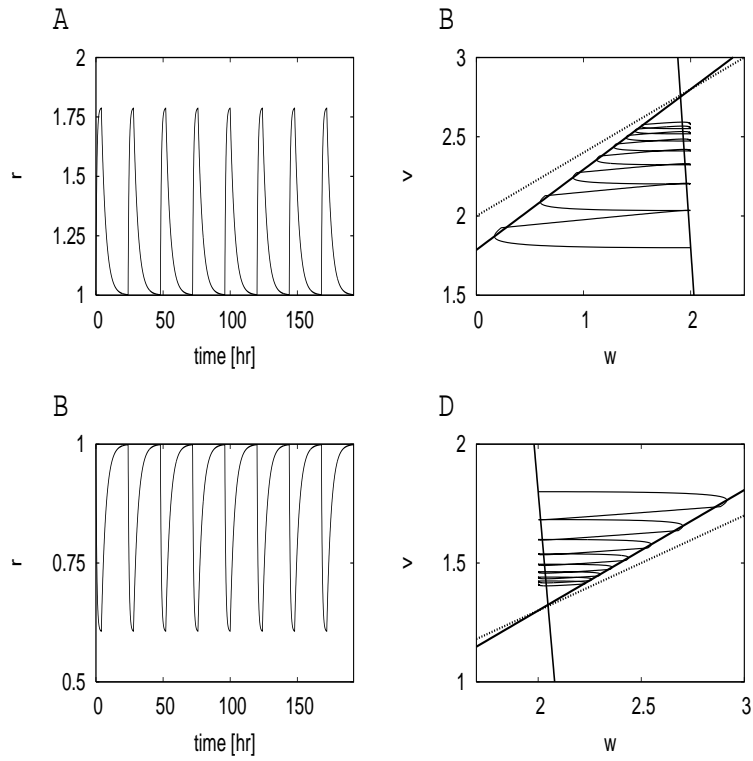

Figure 3: Numerical simulations of savings with the Purkinje cell-dependent learning rule. We set $A = 0.4$, $u = 1$, $w_0 = 2$, $r_0 = 1$, $v_0 = r_0 + Aw_0$, $\eta_1 = 7$, $\eta_3 = 0.3$, $\eta_4 = 0.05$, $\eta_6 = 0.002$. The target gains are (A, B) $r = 2$ and (C, D) $r = 0.5$. (A) and (C) show VOR gains. (B) and (D) show trajectories in the $w$-$v$ space (thin solid lines) together with the nullclines (thick solid lines) and $e = 0$ (thick dotted lines).

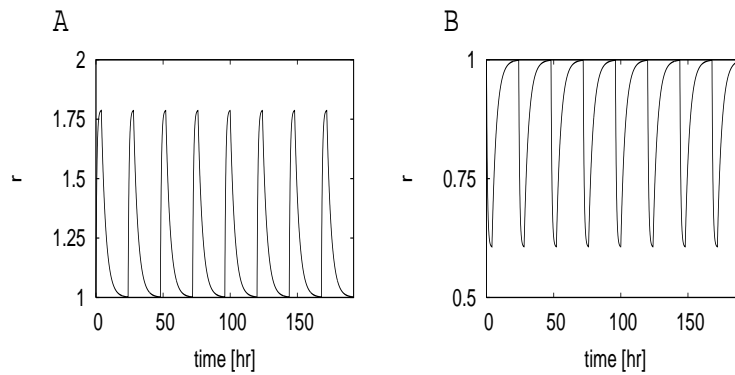

Figure 4: Numerical simulations of savings with fixed $v$. The parameter values are the same as those used in Fig. 3. The target gains are (A) $r = 2$ and (B) $r = 0.5$.

In the earlier models [10, 11], quantitative meanings were given to the equilibrium synaptic weights. Actually, they are solely determined from non-experimentally determined parameters, namely, the balance between the learning rates (in our terminology, $\eta_1$, $\eta_2$, $\eta_4$ and $\eta_5$). Also, the balance seems to play a role in preventing runaway of synaptic weights. In contrast, our model uses the ratio of learning rates (and values of other parameters) just for qualitative purposes and is capable of explaning and predicting experimental settings without parameter tuning. For example, the earlier arguments negating the flocculus hypothesis are based on the fact that the plasticity of the flocculus ($\mathbf{w}$) responding to vestibular inputs occurs but in the direction opposite to the expectation of the flocculus hypothesis [5, 12]. However, this experimental observation is not necessarily contradictory to either the flocculus hypothesis or the two-site hypothesis. As shown in Fig. 2(A), when adapting to a large VOR gain, $\mathbf{w}$ experiences LTD in the initial stage [6]. Then, partial LTP ensues as the motor memory is transferred to the nuclei. Another prediction is about adaptation to a small gain. Figure 2(B) predicts that, in this case, LTP in the indirect pathway is gradually transferred to LTD in the direct pathway. Partial LTD following LTP is anticipated in the flocculus. This implies savings in unlearning.

## Acknowledgments

We thank S. Nagao for helpful discussions. This work was supported by the Special Post-doctoral Researchers Program of RIKEN.

## References

[1] C. D. Aizenman, D. J. Linden. Rapid, synaptically driven increases in the intrinsic excitability of cerebellar deep nuclear neurons. *Nat. Neurosci., 3*, 109–111 (2000).

[2] J. S. Albus. A theory of cerebellar function. *Math. Biosci., 10*, 25–61 (1971).

[3] E. S. Boyden, A. Katoh, J. L. Raymond. Cerebellum-dependent learning: the role of multiple plasticity mechanisms. *Annu. Rev. Neurosci., 27*, 581–609 (2004).

[4] P. Dayan, L. F. Abbott. Theoretial Neuroscience — Computational and Mathematical Modeling of Neural Systems. MIT (2001).

[5] S. du Lac, J. L. Raymond, T. J. Sejnowski, S. G. Lisberger. Learning and memory in the vestibulo-ocular reflex. *Annu. Rev. Neurosci., 18*, 409–441 (1995).

[6] M. Ito. Long-term depression. *Ann. Rev. Neurosci., 12*, 85–102 (1989).

[7] A. E. Luebke, D. A. Robinson. Gain changes of the cat's vestibulo-ocular reflex after flocculus deactivation. *Exp. Brain Res., 98*, 379–390 (1994).

[8] D. Marr. A theory of cerebellar cortex. *J. Physiol., 202*, 437–470 (1969).

[9] M. D. Mauk. Roles of cerebellar cortex and nuclei in motor learning: contradictions or clues? *Neuron, 18*, 343–346 (1997).

[10] J. F. Medina, M. D. Mauk. Simulations of cerebellar motor learning: computational analysis of plasticity at the mossy fiber to deep nucleus synapse. *J. Neurosci., 19*, 7140–7151 (1999).

[11] J. F. Medina, K. S. Garcia, M. D. Mauk. A mechanism for savings in the cerebellum. *J. Neurosci., 21*, 4081–4089 (2001).

[12] F. A. Miles, D. J. Braitman, B. M. Dow. Long-term adaptive changes in primate vestibuloocular reflex. IV. Electrophysiological observations in flocculus of adapted monkeys. *J. Neurophysiol., 43*, 1477–1493 (1980).

[13] B. W. Peterson, J. F. Baker, J. C. Houk. A model of adaptive control of vestibuloocular reflex based on properties of cross-axis adaptation. *Ann. New York Acad. Sci. 627*, 319–337 (1991).

[14] R. J. Racine, D. A. Wilson, R. Gingell, D. Sunderland. Long-term potentiation in the interpositus and vestibular nuclei in the rat. *Exp. Brain Res., 63*, 158–162 (1986).
